# MDPs with Non-Deterministic Policies

**Mahdi Milani Fard**
School of Computer Science
McGill University
Montreal, Canada
mmilan1@cs.mcgill.ca

**Joelle Pineau**
School of Computer Science
McGill University
Montreal, Canada
jpineau@cs.mcgill.ca

## Abstract

Markov Decision Processes (MDPs) have been extensively studied and used in the context of planning and decision-making, and many methods exist to find the optimal policy for problems modelled as MDPs. Although finding the optimal policy is sufficient in many domains, in certain applications such as decision support systems where the policy is executed by a human (rather than a machine), finding all possible near-optimal policies might be useful as it provides more flexibility to the person executing the policy. In this paper we introduce the new concept of *non-deterministic* MDP policies, and address the question of finding near-optimal non-deterministic policies. We propose two solutions to this problem, one based on a Mixed Integer Program and the other one based on a search algorithm. We include experimental results obtained from applying this framework to optimize treatment choices in the context of a medical decision support system.

## 1 Introduction

Markov Decision Processes (MDPs) have been extensively studied in the context of planning and decision-making. In particular, MDPs have emerged as a useful framework for optimizing action choices in the context of medical decision support systems [1, 2, 3, 4]. Given an adequate MDP model (or data source), many methods can be used to find a good action-selection policy. This policy is usually a *deterministic* or *stochastic* function [5]. But policies of these types face a substantial barrier in terms of gaining acceptance from the medical community, because they are highly prescriptive and leave little room for the doctor's input. In such cases, where the actions are executed by a human, it may be preferable to instead provide several (near-)equivalently good action choices, so that the agent can pick among those according to his or her own heuristics and preferences. [1]

To address this problem, this paper introduces the notion of a *non-deterministic policy* [2], which is a function mapping each state to a set of actions, from which the acting agent can choose. We aim for this set to be as large as possible, to provide freedom of choice to the agent, while excluding any action that is significantly worse than optimal. Unlike stochastic policies, here we make no assumptions regarding which action will be executed. This choice can be based on the doctor's qualitative assessment, patient's preferences, or availability of treatment.

While working with non-deterministic policies, it is important to ensure that by adding some freedom of choice to the policy, the worst-case expected return of the policy is still close enough to the optimal value. We address this point by providing guarantees on the expected return of the non-deterministic policy. We define a set of optimization problems to find such a policy and provide two algorithms to solve this problem. The first is based on a Mixed Integer Program formulation, which provides the best solution—in the sense of maximizing the choice of action, while remaining

within an allowed performance-loss threshold—but with high computational cost. Then we describe a simple search algorithm that can be much more efficient in some cases.

The main contributions of this work are to introduce the concept of non-deterministic policies, provide solution methods to compute such policies, and demonstrate the usefulness of this new model for providing acceptable solutions in medical decision support systems. From a pratical perspective, we aim to improve the acceptability of MDP-based decision-support systems.

## 2  Non-Deterministic Policies

In this section, we formulate the concept of non-deterministic policies and provide some definitions that are used throughout the paper.

An **MDP** $M = (S, A, T, R)$ is defined by a set of states $S$, a function $A(s)$ mapping each state to a set of action, a transition function $T(s, a, s')$ defined as:

$$T(s, a, s') = p(s_{t+1} = s' | s_t = s, a_t = a), \forall s, s' \in S, a \in A(s), \tag{1}$$

and a reward function $R(s, a) : S \times A \rightarrow [R_{min}, R_{max}]$. Throughout the paper we assume finite state, finite action, discounted reward MDPs, with the discount factor denoted by $\gamma$.

A **deterministic policy** is a function from states to actions. The **optimal deterministic policy** is the policy that maximizes the expected discounted sum of rewards ($\sum_t \gamma^t r_t$) if the agent acts according to that policy. The **value of a state-action pair** $(s, a)$ according to the optimal deterministic policy on an MDP $M = (S, A, T, R)$ satisfies the Bellman optimality equation [6]:

$$Q_M^*(s, a) = R(s, a) + \gamma \sum_{s'} \left( T(s, a, s') \max_{a' \in A(s')} Q_M^*(s', a') \right). \tag{2}$$

We further define the optimal value of state $s$ denoted by $V_M^*(s)$ to be $\max_{a \in A(s)} Q_M^*(s, a)$.

A **non-deterministic policy** is a function that maps each state $s$ to a non-empty set of actions denoted by $\Pi(s) \subseteq A(s)$. The agent can choose to do any action $a \in \Pi(s)$ whenever the MDP is in state $s$. Here we will provide a worst-case analysis, presuming that the agent may choose the worst action in each state.

The **value of a state-action pair** $(s, a)$ according to a *non-deterministic policy* $\Pi$ on an MDP $M = (S, A, T, R)$ is given by the recursive definition:

$$Q_M^\Pi(s, a) = R(s, a) + \gamma \sum_{s'} \left( T(s, a, s') \min_{a' \in \Pi(s')} Q_M^\Pi(s', a') \right), \tag{3}$$

which is the worst-case expected return under the allowed set of actions. We define the value of state $s$ according to a non-deterministic policy $\Pi$ denoted by $V_M^\Pi(s)$ to be $\min_{a \in \Pi(s)} Q_M^\Pi(s, a)$.

To calculate the value of a non-deterministic policy, we construct an MDP $M' = (S', A', R', T')$ where $S' = S$, $A' = \Pi$, $R' = -R$ and $T' = T$. It is straight-forward to show that:

$$Q_M^\Pi(s, a) = -Q_{M'}^*(s, a). \tag{4}$$

A non-deterministic policy $\Pi$ is said to be **augmented** with state-action pair $(s, a)$ denoted by $\Pi' = \Pi + (s, a)$, if it satisfies:

$$\Pi'(s') = \begin{cases} \Pi(s'), & s' \neq s \\ \Pi(s') \cup \{a\}, & s' = s \end{cases} \tag{5}$$

If a policy $\Pi$ can be achieved by a number of augmentations from a policy $\Pi'$, we say that $\Pi$ includes $\Pi'$. The size of a policy $\Pi$, denoted by $|\Pi|$, is the sum of the cardinality of the action sets in $\Pi$: $|\Pi| = \sum_s |\Pi(s)|$.

A non-deterministic policy $\Pi$ is said to be **non-augmentable** according to a constraint $\Psi$ if and only if $\Pi$ satisfies $\Psi$, and for any state-action pair $(s, a)$, $\Pi + (s, a)$ does not satisfy $\Psi$. In this paper we

will be working with constraints that have this particular property: if a policy $\Pi$ does not satisfy $\Psi$, any policy that includes $\Pi$ does not satisfy $\Psi$. We will refer to such constraints as being **monotonic**.

A non-deterministic policy $\Pi$ on an MDP $M$ is said to be $\epsilon$-**optimal** ($\epsilon \in [0, 1]$) if we have:[3]

$$V_M^\Pi(s) \geq (1 - \epsilon)V_M^*(s), \quad \forall s \in S. \tag{6}$$

This can be thought of as a constraint $\Psi$ on the space of non-deterministic policies which makes sure that the worst-case expected return is within some range of the optimal value. It is straight forward to show that this constraint is monotonic.

A **conservative $\epsilon$-optimal non-deterministic policy** $\Pi$ on an MDP $M$ is a policy that is non-augmentable according to this constraint:

$$R(s, a) + \gamma \sum_{s'} \left( T(s, a, s')(1 - \epsilon)V_M^*(s') \right) \geq (1 - \epsilon)V_M^*(s), \quad \forall a \in \Pi(s). \tag{7}$$

This constraint indicates that we only add those actions to the policy whose reward plus $(1 - \epsilon)$ of the future optimal return is within the sub-optimal margin. This ensures that non-deterministic policy is $\epsilon$-optimal by using the inequality:

$$Q_M^\Pi(s, a) \geq R(s, a) + \gamma \sum_{s'} \left( T(s, a, s')(1 - \epsilon)V_M^*(s') \right), \tag{8}$$

instead of solving Eqn 3 and using the inequality constraint in Eqn 6. Applying Eqn 7 guarantees that the non-deterministic policy is $\epsilon$-optimal while it may still be augmentable according to Eqn 6, hence the name conservative. It can also be shown that the conservative policy is unique.

A **non-augmentable $\epsilon$-optimal non-deterministic policy** $\Pi$ on an MDP $M$ is a policy that is not augmentable according to the constraint in Eqn 6. It is easy to show that any non-augmentable $\epsilon$-optimal policy includes the conservative policy. However, non-augmentable $\epsilon$-optimal policies are not necessarily unique. In this paper we will focus on a search problem in the space of non-augmentable $\epsilon$-optimal policies, trying to maximize some criteria. Specifically, we will be trying to find non-deterministic policies that give the acting agent more options while staying within an acceptable sub-optimal margin.

We now present an example that clarifies the concepts introduced so far. To simplify drawing graphs of the MDP and policies, we assume deterministic transitions in this example. However the concepts apply to any probabilistic MDP as well. Fig 1 shows a sample MDP. The labels on the arcs show action names and the corresponding rewards are shown in the parentheses. We assume $\gamma \simeq 1$ and $\epsilon = 0.05$. Fig 2 shows the optimal policy of this MDP. The conservative $\epsilon$-optimal non-deterministic policy of this MDP is shown in Fig 3.

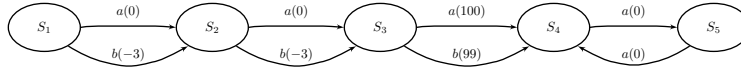

Figure 1: Example MDP

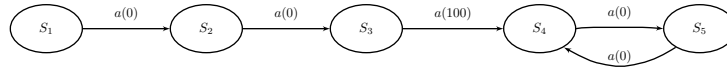

Figure 2: Optimal policy

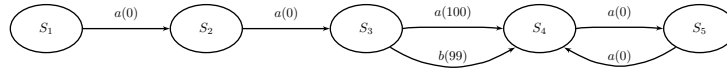

Figure 3: Conservative policy

Fig 4 includes two possible non-augmentable $\epsilon$-optimal policies. Although both policies in Fig 4 are $\epsilon$-optimal, the union of these is not $\epsilon$-optimal. This is due to the fact that adding an option to one of the states removes the possibility of adding options to other states, which illustrates why local changes are not always appropriate when searching in the space of $\epsilon$-optimal policies.

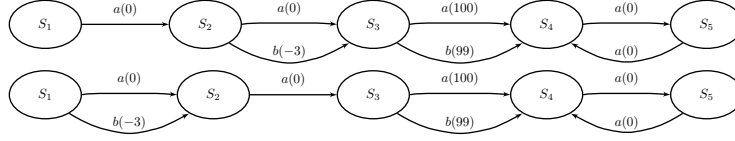

Figure 4: Two non-augmentable policies

## 3 Optimization Problem

We formalize the problem of finding an $\epsilon$-optimal non-deterministic policy in terms of an optimization problem. There are several optimization criteria that can be formulated, while still complying with the $\epsilon$-optimal constraint. Notice that the last two problems can be defined both in the space of all $\epsilon$-optimal policies or only the non-augmentable ones.

- **Maximizing the size of the policy**: According to this criterion, we seek non-augmentable $\epsilon$-optimal policies that have the biggest overall size. This provides more options to the agent while still keeping the $\epsilon$-optimal guarantees. The algorithms proposed in this paper use this optimization criterion. Notice that the solution to this optimization problem is non-augmentable according to the $\epsilon$-optimal constraint, because it maximizes the overall size of the policy.

- **Maximizing the margin**: We aim to maximize **margin of a non-deterministic policy** $\Pi$:

$$\Phi_M(\Pi) = \min_s \left( \min_{a \in \Pi(s), a' \notin \Pi(s)} (Q(s,a) - Q(s,a')) \right). \tag{9}$$

This optimization criterion is useful when one wants to find a clear separation between the good and bad actions in each state.

- **Minimizing the uncertainly**: If we learn the models from data we will have some uncertainly about the optimal action in each state. We can use some variance estimation on the value function [8] along with a Z-Test to get some confidence level on our comparisons and find the probability of having the wrong order when comparing actions according to their values. Let $Q$ be the value of the true model and $\hat{Q}$ be our empirical estimate based on some dataset $D$. We aim to minimize the **uncertainly of a non-deterministic policy** $\Pi$:

$$\Phi_M(\Pi) = \max_s \left( \max_{a \in \Pi(s), a' \notin \Pi(s)} p \left( Q(s,a) < Q(s,a') | D \right) \right). \tag{10}$$

## 4 Solving the Optimization Problem

In the following sections we provide algorithms to solve the first optimization problem mentioned above, which aims to maximize the size of the policy. We focus on this criterion as it seems most appropriate for medical decision support systems, where it is desirable for the acceptability of the system to find policies that provide as much choice as possible for the acting agent. We first present a Mixed Integer Program formulation of the problem, and then present a search algorithm that uses the monotonic property of the $\epsilon$-optimal constraint. While the MIP method is useful as a general formulation of the problem, the search algorithm has potential for further extensions with heuristics.

### 4.1 Mixed Integer Program

Recall that we can formulate the problem of finding the optimal deterministic policy on an MDP as a simple linear program [5]:

$$\min_V \mu^T V, \text{ subject to}$$
$$V(s) \geq R(s,a) + \gamma \sum_{s'} T(s,a,s')V(s') \quad \forall s,a, \tag{11}$$

where $\mu$ can be thought of as the initial distribution over the states. The solution to the above problem is the optimal value function (denoted by $V^*$). Similarly, having computed $V^*$ using Eqn 11, the

problem of a search for an optimal non-deterministic policy according to the size criterion can be rewritten as a Mixed Integer Program:[4]

$$\max_{V,\Pi}(\mu^T V + (V_{max} - V_{min})e_s^T \Pi e_a), \text{ subject to}$$
$$V(s) \geq (1-\epsilon)V^*(s) \qquad \forall s$$
$$\sum_a \Pi(s,a) > 0 \qquad \forall s$$
$$V(s) \leq R(s,a) + \gamma \sum_{s'} T(s,a,s')V(s') + V_{max}(1-\Pi(s,a)) \quad \forall s,a. \qquad (12)$$

Here we are overloading the notation $\Pi$ to define a binary matrix representing the policy. $\Pi(s,a)$ is 1 if $a \in \Pi(s)$, and 0 otherwise. We define $V_{max} = R_{max}/(1-\gamma)$ and $V_{min} = R_{min}/(1-\gamma)$. $e$'s are column vectors of 1 with the appropriate dimensions. The first set of constraints makes sure that we stay within $\epsilon$ of the optimal return. The second set of constraints ensures that at least one action is selected per state. The third set ensures that for those state-action pairs that are chosen in any policy, the Bellman constraint holds, and otherwise, the constant $V_{max}$ makes the constraint trivial. Notice that the solution to the above maximizes $|\Pi|$ and the result is non-augmentable. As a counter argument, suppose that we could add a state-action pair to the solution $\Pi$, while still staying in $\epsilon$ sub-optimal margin. By adding that pair, the objective function is increased by $(V_{max} - V_{min})$, which is bigger than any possible decrease in the $\mu^T V$ term, and thus the objective is improved, which conflicts with $\Pi$ being the solution.

We can use any MIP solver to solve the above problem. Note however that we do not make use of the monotonic nature of the constraints. A general purpose MIP solver could end up searching in the space of all the possible non-deterministic policies, which would require exponential running time.

## 4.2 Search Algorithm

We can make use of the monotonic property of the $\epsilon$-optimal policies to narrow down the search. We start by computing the conservative policy. We then augment it until we arrive at a non-augmentable policy. We make use of the fact that if a policy is not $\epsilon$-optimal, neither is any other policy that includes it, and thus we can cut the search tree at this point.

The following algorithm is a one-sided recursive depth-first-search-like algorithm that searches in the space of plausible non-deterministic policies to maximize a function $g(\Pi)$. Here we assume that there is an ordering on the set of state-action pairs $\{p_i\} = \{(s_j, a_k)\}$. This ordering can be chosen according to some heuristic along with a mechanism to cut down some parts of the search space. $V^*$ is the optimal value function and the function $V$ returns the value of the non-deterministic policy that can be calculated by minimizing Equation 3.

**Function** $getOptimal(\Pi, startIndex, \epsilon)$
$\Pi_o \leftarrow \Pi$
**for** $i \leftarrow startIndex$ **to** $|S||A|$ **do**
    $(s,a) \leftarrow p_i$
    **if** $a \notin \Pi(s)$ & $V(\Pi + (s,a)) \geq (1-\epsilon)V^*$ **then**
        $\Pi' \leftarrow getOptimal(\Pi + (s,a), i+1, \epsilon)$
        **if** $g(\Pi') > g(\Pi_o)$ **then**
            $\Pi_o \leftarrow \Pi'$
        **end**
    **end**
**end**
**return** $\Pi_o$

We should make a call to the above function passing in the conservative policy $\Pi_m$ and starting from the first state-action pair: $getOptimal(\Pi_m, 0, \epsilon)$.

The asymptotic running time of the above algorithm is $O((|S||A|)^d(t_m + t_g))$, where $d$ is the maximum size of an $\epsilon$-optimal policy minus the size of the conservative policy, $t_m$ is the time to solve the original MDP and $t_g$ is the time to calculate the function $g$. Although the worst-case running time is still exponential in the number of state-action pairs, the run-time is much less when the search space is sufficiently small. The $|A|$ term is due to the fact that we check all possible augmentations for

each state. Note that this algorithm searches in the space of all $\epsilon$-optimal policies rather than only the non-augmentable ones. If we set function $g(\Pi) = |\Pi|$, then the algorithm will return the biggest non-augmentable $\epsilon$-optimal policy.

This search can be further improved by using heuristics to order the state-action pairs and prune the search. One can also start the search from any other policy rather than the conservative policy. This can be potentially useful if we have further constraints on the problem. One way to narrow down the search is to only add the action that has the maximum value for any state $s$:

$$\Pi' = \Pi + \left( s, \arg \max_{Q(s,a)} \right), \tag{13}$$

This leads to a running time of $O(|S|^d (t_m + t_g))$. However this does not guarantee that we see all non-augmentable policies. This is due to the fact that after adding an action, the order of values might change. If the transition structure of the MDP contains no loop with non-zero probability (transition graph is directed acyclic, DAG), then this heuristic will produce the optimal result while cutting down the search time. In other cases, one might do a partial evaluation of the augmented policy to approximate the value after adding the actions, possibly by doing a few backups rather than using the original $Q$ values. This offers the possibility of trading-off computation time for better solutions.

## 5   Empirical Evaluation

To evaluate our proposed algorithms, we first test the both the MIP and search formulations on MDPs created randomly, and then test the search algorithm on a real-world treatment design scenario.

To begin, we generated random MDPs with 5 states and 4 actions. The transitions are deterministic (chosen uniformly random) and the rewards are random values between 0 and 1, except for one of the states with reward 10 for one of the actions; $\gamma$ was set to 0.95. The MIP method was implemented with MATLAB and CPLEX. Fig 5 shows the solution to the MIP defined in Eqn 12 for a particular randomly generated MDP. We see that the size of non-deterministic policy increases as the performance threshold is relaxed.

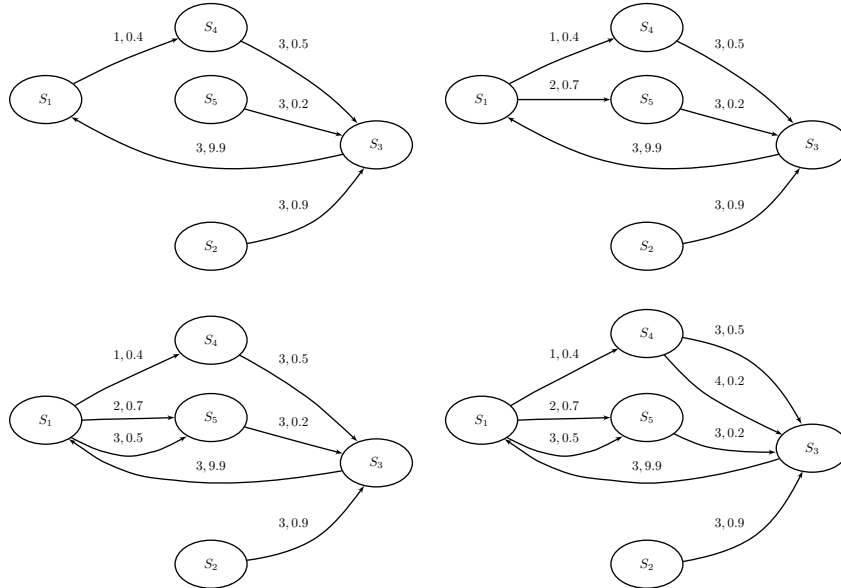

Figure 5: MIP solution for different values of $\epsilon \in \{0, 0.01, 0.02, 0.03\}$. The labels on the edges are action indices, followed by the corresponding immediate rewards.

To compare the running time of the MIP solver and the search algorithm, we constructed random MDPs as described above with more state-action pairs. Fig 6 Left shows the running time averaged over 20 different random MDPs , assuming $\epsilon = 0.01$. It can be seen that both algorithms have

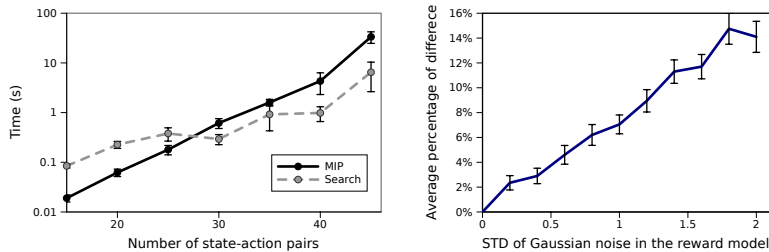

Figure 6: Left: Running time of MIP and search algorithm as a function of the number of state-action pairs. Right: Average percentage of state-action pairs that were different in the noisy policy.

exponential running time. The running time of the search algorithm has a bigger constant factor, but has a smaller exponent base which results in a faster asymptotic running time.

To study how stable non-deterministic policies are to potential noise in the models, we check to see how much the policy changes when Gaussian noise is added to the reward function. Fig 6 Right shows the percentage of the total state-action pairs that were either added or removed from the resulting policy by adding noise to the reward model (we assume a constant $\epsilon = 0.02$). We see that the resulting non-deterministic policy changes somewhat, but not drastically, even with noise level of similar magnitude as the reward function.

Next, we implemented the full search algorithm on an MDP constructed for a medical decision-making task involving real patient data. The data was collected as part of a large (4000+ patients) multi-step randomized clinical trial, designed to investigate the comparative effectiveness of different treatments provided sequentially for patients suffering from depression [9]. The goal is to find a treatment plan that maximizes the chance of remission. The dataset includes a large number of measured outcomes. For the current experiment, we focus on a numerical score called the Quick Inventory of Depressive Symptomatology (QIDS), which was used in the study to assess levels of depression (including when patients achieved remission). For the purposes of our experiment, we discretize the QIDS scores (which range from 5 to 27) uniformly into quartiles, and assume that this, along with the treatment step (up to 4 steps were allowed), completely describe the patient's *state*. Note that the underlying transition graph can be treated as a DAG because the study is limited to four steps of treatment. There are 19 actions (treatments) in total. A reward of 1 is given if the patient achieves remission (at any step) and a reward of 0 is given otherwise. The transition and reward models were generated empirically from the data using a frequentist approach.

Table 1: Policy and running time of the full search algorithm on the medical problem

|  | $\epsilon = 0.02$ | $\epsilon = 0.015$ | $\epsilon = 0.01$ | $\epsilon = 0$ |
|---|---|---|---|---|
| Time (seconds) | 118.7 | 12.3 | 3.5 | 1.4 |
| $5 < QIDS < 9$ | CT<br>SER<br>BUP, CIT+BUS | CT<br>SER | CT | CT |
| $9 \leq QIDS < 12$ | CIT+BUP<br>CIT+CT | CIT+BUP<br>CIT+CT | CIT+BUP | CIT+BUP |
| $12 \leq QIDS < 16$ | VEN<br>CIT+BUS<br>CT | VEN<br>CIT+BUS | VEN | VEN |
| $16 \leq QIDS \leq 27$ | CT<br>CIT+CT | CT<br>CIT+CT | CT<br>CIT+CT | CT |

Table 1 shows the non-deterministic policy obtained for each state during the second step of the trial (each acronym refers to a specific treatment). This is computed using the search algorithm, assuming different values of $\epsilon$. Although this problem is not tractable with the MIP formulation (304 state-action pairs), a full search in the space of $\epsilon$-optimal policies is still possible. Table 1 also shows the running time of the algorithm, which as expected increases as we relax the threshold $\epsilon$. Here we did not use any heuristics. However, as the underlying transition graph is a DAG, we could use the heuristic discussed in the previous section (Eqn 13) to get the same policies even faster. An

interesting question is how to set $\epsilon$ a priori. In practice, a doctor may use the full table as a guideline, using smaller values of $\epsilon$ when s/he wants to rely more on the decision support system, and larger values when relying more on his/her own assessments.

## 6    Discussion

This paper introduces a framework for computing non-deterministic policies for MDPs. We believe this framework can be especially useful in the context of decision support systems to provide more choice and flexibility to the acting agent. This should improve acceptability of decision support systems in fields where the policy is used to guide (or advise) a human expert, notably for the optimization of medical treatments.

The framework we propose relies on two competing objectives. On the one hand we want to provide as much choice as possible in the non-deterministic policy, while at the same time preserving some guarantees on the return (compared to the optimal policy). We present two algorithms that can solve such an optimization problem: a MIP formulation that can be solved by any general MIP solver, and a search algorithm that uses the monotonic property of the studied constraints to cut down on the running time. The search algorithm is particularly useful when we have good heuristics to further prune the search space. Future work will consider different optimizing criteria, such as those outlined in Section 3, which may be more appropriate for some domains with very large action sets.

A limitation of our current approach is that the algorithms presented so far are limited to relatively small domains, and scale well only for domains with special properties, such as a DAG structure in the transition model or good heuristics for pruning the search. This clearly points to future work in developing better approximation techniques. Nonetheless it is worth keeping in mind that many domains of application, may not be that large (see [1, 2, 3, 4] for examples) and the techniques as presented can already have a substantial impact.

Finally, it is worth noting that non-deterministic policies can also be useful in cases where the MDP transition and reward models are imperfectly specified or learned from data, though we have not explored this case in detail yet. In such a setting, the difference between the optimal and a near optimal policy may not be computed accurately. Thus, it is useful to find all actions that are close to optimal so that the real optimal action is not missed. An interesting question here is whether we can find the smallest non-deterministic policy that will include the optimal policy with some probability $1 - \delta$. This is similar to the framework in [7], and could be useful in cases where there is not enough data to compare policies with good statistical significance.

**Acknowledgements:** The authors wish to thank A. John Rush, Susan A. Murphy, Doina Precup, and Stephane Ross for helpful discussions regarding this work. Funding was provided by the National Institutes of Health (grant R21 DA019800) and the NSERC Discovery Grant program.

## Footnotes

[1] This is especially useful given that human preferences are often difficult to quantify objectively, and thus difficult to incorporate in the reward function.

[2] Borrowing the term "non-deterministic" from the theory of computation, as opposed to deterministic or stochastic actions.

[3]In some of the MDP literature, $\epsilon$-optimality is defined as an additive constraint ($Q_M^\Pi \geq Q_M^* - \epsilon$) [7]. The derivations will be analogous in that case.

[4]Note that in this MIP, unlike the standard LP for MDPs, the choice of $\mu$ can affect the solution in cases where there is a tie in the size of $\Pi$.

## References

[1] A. Schaefer, M. Bailey, S. Shechter, and M. Roberts. *Handbook of Operations Research / Management Science Applications in Health Care*, chapter Medical decisions using Markov decision processes. Kluwer Academic Publishers, 2004.

[2] M. Hauskrecht and H. Fraser. Planning treatment of ischemic heart disease with partially observable Markov decision processes. *Artificial Intelligence in Medicine*, 18(3):221–244, 2000.

[3] P. Magni, S. Quaglini, M. Marchetti, and G. Barosi. Deciding when to intervene: a Markov decision process approach. *International Journal of Medical Informatics*, 60(3):237–253, 2000.

[4] D. Ernst, G. B. Stan, J. Concalves, and L. Wehenkel. Clinical data based optimal sti strategies for hiv: a reinforcement learning approach. In *Proceedings of Benelearn*, 2006.

[5] D.P. Bertsekas. *Dynamic Programming and Optimal Control, Vol 2*. Athena Scientific, 1995.

[6] R.S. Sutton and A.G. Barto. *Reinforcement Learning: An Introduction*. MIT Press, Cambridge, MA, 1998.

[7] M. Kearns and S. Singh. Near-optimal reinforcement learning in poly. time. *Machine Learning*, 49, 2002.

[8] S. Mannor, D. Simester, P. Sun, and J.N. Tsitsiklis. Bias and variance in value function estimation. In *Proceedings of ICML*, 2004.

[9] M. Fava, A.J. Rush, and M.H. Trivedi et al. Background and rationale for the sequenced treatment alternatives to relieve depression (STAR*D) study. *Psychiatr Clin North Am*, 26(2):457–94, 2003.